# On the concentration of expectation and approximate inference in layered networks

**XuanLong Nguyen**
University of California
Berkeley, CA 94720
xuanlong@cs.berkeley.edu

**Michael I. Jordan**
University of California
Berkeley, CA 94720
jordan@cs.berkeley.edu

## Abstract

We present an analysis of concentration-of-expectation phenomena in layered Bayesian networks that use generalized linear models as the local conditional probabilities. This framework encompasses a wide variety of probability distributions, including both discrete and continuous random variables. We utilize ideas from large deviation analysis and the delta method to devise and evaluate a class of approximate inference algorithms for layered Bayesian networks that have superior asymptotic error bounds and very fast computation time.

## 1 Introduction

The methodology of variational inference has developed rapidly in recent years, with increasingly rich classes of approximation being considered (see, e.g., Yedidia, et al., 2001, Jordan et al., 1998). While such methods are intuitively reasonable and often perform well in practice, it is unfortunately not possible, except in very special cases, to provide error bounds for these inference algorithms. Thus the user has little a priori guidance in choosing an inference algorithm, and little a posteriori reassurance that the approximate marginals produced by an algorithm are good approximations. The situation is somewhat better for sampling algorithms, but there the reassurance is only asymptotic.

A line of research initiated by Kearns and Saul (1998) aimed at providing such error bounds for certain classes of directed graphs. Analyzing the setting of two-layer networks, binary nodes with large fan-in, noisy-OR or logistic conditional probabilities, and parameters that scale as $O(1/N)$, where $N$ are the number of nodes in each layer, they used a simple large deviation analysis to design an approximate inference algorithm that provided error bounds. In later work they extended their algorithm to multi-layer networks (Kearns and Saul, 1999). The error bound provided by this approach was $O(\sqrt{\ln N/N})$. Ng and Jordan (2000) pursued this line of work, obtaining an improved error bound of $O(1/N^{(k+1)/2})$ where $k$ is the order of a Taylor expansion employed by their technique. Their approach was, however, restricted to two-layer graphs.

Layered graphs are problematic for many inference algorithms, including belief propagation and generalized belief propagation algorithms. These algorithms convert directed graphs to undirected graphs by moralization, which creates infeasibly large cliques when there are nodes with large fan-in. Thus the work initiated by Kearns and Saul is notable not only for its ability to provide error bounds, but also because it provides one of the few

practical algorithms for general layered graphs. It is essential to develop algorithms that scale in this setting—e.g., a recent application at Google studied layered graphs involving more than a million nodes (Harik and Shazeer, personal communication).

In this paper, we design and analyze approximate inference algorithms for general multi-layered Bayesian networks with generalized linear models as the local conditional probability distributions. Generalized linear models including noisy-OR and logistic functions in the binary case, but go significantly further, allowing random variables from any distribution in the exponential family. We show that in such layered graphical models, the concentration of expectations of any fixed number of nodes propagate from one layer to another according to a topological sort of the nodes. This concentration phenomenon can be exploited to devise efficient approximate inference algorithms that provide error bounds. Specifically, in a multi-layer network with $N$ nodes in each layer and random variables in some exponential family of distribution, our algorithm has an $O((\ln N)^3/N)^{(k+1)/2})$ error bound and $O(N^k)$ time complexity. We perform a large number of simulations to confirm this error bound and compare with Kearns and Saul's algorithm, which has not been empirically evaluated before.

The paper is organized as follows. In Section 2, we study the concentration of expectation in generalized linear models. Section 3 introduces the use of delta method for approximating the expectations. Section 4 describes an approximate inference algorithm in a general directed graphical model, which is evaluated empirically in Section 5. Finally, Section 6 concludes the paper.

## 2  Generalized linear models

Consider a generalized linear model (GLIM; see McCullagh and Nelder, 1983, for details) consisting of $N$ covariates (inputs) $X_1, \ldots, X_N$ and a response (output) variable $Y$. A GLIM makes three assumptions regarding the form of the conditional probability distribution $P(Y|X)$: (1) The inputs $X_1, \ldots, X_N$ enter the model via a linear combination $\xi = \sum_{i=1}^N \theta_i X_i$; (2) the conditional mean $\mu$ is represented as a function $f(\xi)$, known as the response function; and (3) the output $Y$ is characterized by an exponential family distribution (cf. Brown, 1986) with natural parameter $\eta$ and conditional mean $\mu$. The conditional probability takes the following form:

$$P_{\theta,\phi}(Y|X) = h(y,\phi) \exp \frac{\eta y - A(\eta)}{\phi}, \tag{1}$$

where $\phi$ is a scale parameter, $h$ is a function reflecting the underlying measure, and $A(\eta)$ is the log partition function.

In this section, for ease of exposition, we shall assume that the response function $f$ is a canonical response function, which simply means that $\eta = \xi = \sum_{i=1}^N \theta_i X_i$. As will soon be clear, however, our analysis is applicable to a general setting in which $f$ is only required to have bounded derivatives on compact sets.

It is a well-known property of exponential family distributions that

$$E(Y|X) \;=\; \mu = A'(\eta) = f(\eta) = f\left(\sum_{i=1}^N \theta_i X_i\right)$$

$$\mathrm{Var(Y|X)} \;=\; \phi A''(\eta) = \phi f'(\eta).$$

The exponential family includes the Bernoulli, multinomial, and Gaussian distributions, but many other useful distributions as well, including the Poisson, gamma and Dirichlet.

We will be studying GLIMs defined on layered graphical models, and thus $X_1, \ldots, X_N$ are themselves taken to be random variables in the exponential family. We also make the key

assumption that all parameters obey the bound $|\theta_i| \leq \tau/N$ for some constant $\tau$, although this assumption shall be relaxed later on.

Under these assumptions, we can show that the linear combination $\eta = \sum_{i=1}^{N} \theta_i X_i$ is tightly concentrated around its mean with very high probability. Kearns and Saul (1998) have proved this for binary random variables using large deviation analysis. This type of analysis can be used to prove general results for (bounded and unbounded) random variables in any standard exponential family.[1]

**Lemma 1** *Assume that $X_1, \ldots, X_N$ are independent random variables in a standard exponential family distribution. Furthermore, $EX_i \in [p_i - \Delta_i, p_i + \Delta_i]$. Then there are absolute constants $C$ and $\alpha$ such that, for any $\epsilon > \sum_{i=1}^{N} |\theta_i| \Delta_i$:*

$$P(|\eta - \sum_{i=1}^{N} \theta_i p_i| > \epsilon) \leq C \exp - \frac{\alpha(\epsilon - \sum_{i=1}^{N} |\theta_i| \Delta_i)^{2/3}}{(\sum_{i=1}^{N} \theta_i^2)^{1/3}}$$

$$\leq C \exp\{-\alpha N^{1/3} \tau^{-2/3}(\epsilon - \sum_{i=1}^{N} |\theta_i| \Delta_i)^{2/3}\}$$

We will study architectures that are strictly layered; that is, we require that there are no edges directly linking the parents of any node. In this setting the parents of each node are conditionally independent given all ancestor nodes (in the previous layers) in the graph. This will allow us to use Lemma 1 and iterated conditional expectation formulas to analyze concentration phenomena in these models. The next lemma shows that under certain assumptions about the response function $f$, the tight concentration of $\eta$ also entails the concentration of $E(Y|X)$ and $\mathrm{Var}(Y|X)$.

**Lemma 2** *Assume that the means of $X_1, \ldots, X_N$ are bounded within some fixed interval $[p_{min}, p_{max}]$ and $f$ has bounded derivatives on compact sets. If $\eta \in [\sum_{i=1}^{N} \theta_i p_i - \epsilon, \sum_{i=1}^{N} \theta_i p_i + \epsilon]$ with high probability, then: $E(Y|X) = f(\eta) \in [f(\sum_{i=1}^{N} \theta_i p_i) - O(\epsilon), f(\sum_{i=1}^{N} \theta_i p_i) + O(\epsilon)]$, and $\mathrm{Var}(Y|X) = f'(\eta) \in [f'(\sum_{i=1}^{N} \theta_i p_i) - O(\epsilon), f'(\sum_{i=1}^{N} \theta_i p_i) + O(\epsilon)]$ with high probability.*

Lemmas 1 and 2 provide a mean-field-like basis for propagating the concentration of expectations from the input layer $X_1, \ldots, X_N$ to the output layer $Y$. Specifically, if $E(X_i)$ are approximated by $p_i$ ($i = 1, \ldots, N$), then $E(Y)$ can be approximated by $f(\sum_{i=1}^{N} \theta_i p_i)$.

## 3  Higher order expansion (the delta method)

While Lemmas 1 and 2 already provide a procedure for approximating $E(Y)$, one can use higher-order (Taylor) expansion to obtain a significantly more accurate approximation. This approach, known in the statistics literature as the delta method, has been used in slightly different contexts for inference problems in the work of Plefka (1982), Barber and van der Laar (1999), and Ng and Jordan (2000). In our present setting, we will show that estimates based on Taylor expansion up to order $k$ can be obtained by propagating the expectation of the product of up to $k$ nodes from one layer to an offspring layer.

The delta method is based on the same assumptions as in Lemma 2; that is, the means of $X_1, \ldots, X_N$ are assumed to be bounded within some fixed interval $[p_{min}, p_{max}]$, and the response function $f$ has bounded derivatives on compact sets. We have $\sum_{i=1}^{N} \theta_i p_i$ bounded within fixed interval $[\tau p_{min}, \tau p_{max}]$. By Lemma 1, with high probability $\eta =$

$\sum_{i=1}^{N} \theta_i p_i + \epsilon$, for some small $\epsilon$. Using Taylor's expansion up to second order, we have that with high probability:

$$E(Y) \quad = \quad E_x E(Y|X) = E_x f(\eta) = f_\eta +$$

$$(\sum_{i=1}^{N} \theta_i E X_i - \sum_{i=1}^{N} \theta_i p_i) f'_\eta + \frac{1}{2!} (\sum_{i,j} \theta_i \theta_j (E(X_i - p_i)(X_j - p_j))) f''_\eta + O(\epsilon^3),$$

where $f_\eta$ and its derivatives are evaluated at $\sum_{i=1}^{N} \theta_i p_i$. This gives us a method of approximating $E(Y)$ by recursion: Assuming that one can approximate all needed expectations of variables in the parent layer $X$ with error $O(\epsilon^3)$, one can also obtain an approximation of $E(Y)$ with the error $O(\epsilon^3)$. Clearly, the error can be improved to $O(\epsilon^{k+1})$ by using Taylor expansion to some order $k$ (provided that the response function $f(\eta) = A'(\eta)$ has bounded derivatives up to that order). In this case, the expectation of the product of up to $k$ elements in the input layer, e.g., $E(X_1 - p_1) \ldots (X_k - p_k)$, needs to be computed.

The variance of $Y$ (as well as other higher-order expectations) can also be approximated in the same way:

$$\text{Var(Y)} \quad = \quad E_x(\text{Var(Y|X)}) + \text{Var}_x(\text{E(Y|X)})$$

$$= \quad \phi E_x f'(\eta) + E_x f(\eta)^2 - (E(Y))^2$$

where each component can be approximated using the delta method.

## 4  Approximate inference for layered Bayesian networks

In this section, we shall harness the concentration of expectation phenomenon to design and analyze a family of approximate inference algorithms for multi-layer Bayesian networks that use GLIMs as local conditional probabilities. The recipe is clear by now. First, organize the graph into layers that respect the topological ordering of the graph. The algorithm is comprised of two stages: (1) Propagate the concentrated conditional expectations from ancestor layers to offspring layers. This results in a rough approximation of the expectation of individual nodes in the graph; (2) Apply the delta method to obtain more a refined marginal expectation of the needed statistics, also starting from ancestor layers to offspring layers.

Consider a multi-layer network that has $L$ layers, each of which has $N$ random variables. We refer to the $i$th variable in layer $l$ by $X_i^l$, where $\{X_i^1\}_{i=1}^N$ is the input layer, and $\{X_i^L\}_{i=1}^N$ is the output layer. The expectations $E(X_i^1)$ of the first layer are given. For each $2 \leq l \leq L$, let $\theta_{ij}^{l-1}$ denote the parameter linking $X_i^l$ and its parent $X_j^{l-1}$. Define the weighted sum of contributions from parents to a node $X_i^l$: $\eta_i^l = \sum_{j=1}^N \theta_{ij}^{l-1} X_j^{l-1}$, where we assume that $|\theta_{ij}^l| \leq \tau/N$ for some constant $\tau$.

We first consider the problem of estimating expectations of nodes in the output layer. For binary networks, this amounts to estimating marginal probabilities, say, $P[X_1^L = x_1, \ldots, X_m^L = x_m]$, for given observed values $(x_1, \ldots, x_m)$, where $m < N$. We subsequently consider a more general inference problem involving marginal and conditional probabilities of nodes residing in different layers in the graph.

### 4.1  Algorithm stage 1: Propagating the concentrated expectation of single nodes

We establish a rough approximation of the expectations of all single nodes of the graph, starting from the input layer $l = 1$ to the output layer $l = L$ in an inductive manner. For $l = 1$, let $\Delta_i^1 = \delta_i^1 = 0$ and $p_i^1 = E X_i^1$ for all $i = 1, \ldots, N$. For $l > 1$, let

$$\mu_i^l \quad = \quad \sum_{j=1}^N \theta_{ij}^{l-1} p_j^{l-1} \tag{2}$$

$$\epsilon_i^l \quad = \quad \sum_{j=1}^{N} |\theta_{ij}^{l-1}| \Delta_j^{l-1} + \tau \sqrt{(\gamma \ln N)^3 / N} \tag{3}$$

$$\delta_i^l \quad = \quad C \exp\{-\alpha N^{1/3} \tau^{-2/3} (\epsilon_i^l - \sum_{i=1}^{N} |\theta_{ij}^{l-1}| \Delta_j^{l-1})^{2/3}\} \tag{4}$$

$$p_i^l \quad = \quad \frac{1}{2} \left( \sup_{x \in A_i^l} f(x) + \inf_{x \in A_i^l} f(x) \right) \tag{5}$$

$$\Delta_i^l \quad = \quad \frac{1}{2} \left( \sup_{x \in A_i^l} f(x) - \inf_{x \in A_i^l} f(x) \right) \text{ where } A_i^l = [\mu_i^l - \epsilon_i^l, \mu_i^l + \epsilon_i^l]. \tag{6}$$

In the above updates, constants $\alpha$ and $C$ arise from Lemma 1, $\gamma$ is an arbitrary constant that is greater than $1/\alpha$. The following proposition, whose proof makes use of Lemma 1 combined with union bounds, provides the error bounds for our algorithm.

**Proposition 3** *With probability at least $\prod_{l=1}^{L}(1 - \sum_{i=1}^{N} \delta_i^l) = (1 - CN^{1-\alpha\gamma})^{L-1}$, for any $1 \le i \le N, 1 \le l \le L$ we have: $E[X_i^l | X_1^{l-1}, \dots, X_N^{l-1}] = f(\eta_i^l) \in [p_i^l - \Delta_i^l, p_i^l + \Delta_i^l]$ and $\eta_i^l \in [\mu_i^l - \epsilon_i^l, \mu_i^l + \epsilon_i^l]$. Furthermore, $\epsilon_i^l = O(\sqrt{(\ln N)^3/N})$ for all $i, l$.*

For layered networks with only bounded and Gaussian variables, Lemma 1 can be tightened, and this results in an error bound of $O(\sqrt{(\ln N)^2/N})$. For layered networks with only bounded variables, the error bound can be tightened to $O(\sqrt{\ln N/N})$. In addition, if we drop the conditions that all parameters $\theta_{ij}^l$ are bounded by $\tau/N$, Proposition 3 still goes through by replacing $\tau$ by $\sqrt{N \sum_{j=1}^{N} (\theta_{ij}^{l-1})^2}$ in updating equations for $\epsilon_i^l$ and $\delta_i^l$ for all $i$ and $l$. The asymptotic error bound $O(\sqrt{(\ln N)^3/N})$ no longer holds, but it can be shown that there are absolute constants $c_1$ and $c_2$ such that for all $i, l$:

$$\epsilon_i^l \le (c_1 ||\epsilon^{l-1}|| + c_2 \sqrt{(\ln N)^3}) ||\theta_i^{l-1}||$$

where $||\theta_i^{l-1}|| \equiv \sqrt{\sum_{j=1}^{N} (\theta_{ij}^{l-1})^2}$ and $||\epsilon^l|| \equiv \sqrt{\sum_{i=1}^{N} (\epsilon_i^l)^2}$.

## 4.2 Algorithm stage 2: Approximating expectations by recursive delta method

The next step is to apply the delta method presented in Section 3 in a recursive manner. Write:

$$E[X_1^L ... X_m^L] = E_{X^{L-1}} E[X_1^L ... X_m | X^{L-1}] = E_{X^{L-1}} \prod_{i=1}^{m} f(\eta_i^L) = E_{X^{L-1}} F(\eta_1^L, ..., \eta_m^L)$$

where $F(\eta_1^L, ..., \eta_m^L) := \prod_{i=1}^{m} f(\eta_i^L)$.

Let $\beta_i^l = \eta_i^l - \mu_i^l$. So, with probability $(1 - CN^{1-\alpha\gamma})^{L-1}$ we have $|\beta_i^l| \le \epsilon_i^l = O(\sqrt{(\ln N)^3/N})$ for all $l = 1, \dots, L$ and $i = 1, \dots, N$. Applying the delta method by expanding $F$ around the vector $\mu = (\mu_1^L, ..., \mu_m^L)$ up to order $k$ gives an approximation, which is denoted by MF(k), that depends on expectations of nodes in the previous layer. Continuing this approximation recursively on the previous layers, we obtain an approximate algorithm that has an error bound $O(((\ln N)^3/N)^{(k+1)/2})$ (see the derivation in Section 3) with probability at least $(1 - CN^{1-\alpha\gamma})^{L-1}$ and an error bound $O(1)$ with the remaining probability. We conclude that,

**Theorem 4** *The absolute error of the MF(k) approximation is $O(((\ln N)^3/N)^{(k+1)/2})$. For networks with bounded variables, the error bound can be tightened to $O((\ln N/N)^{(k+1)/2})$.*

It is straightforward to check that MF($k$) takes $O(N^{\max\{k,2\}})$ computational time. The asymptotic error bound $O(((\ln N)^3/N)^{(k+1)/2})$ is guaranteed for the aproximation of expectations of a fixed number $m$ of nodes in the output layer. In principle, this implies that $m$ has to be small compared to $N$ for the approximation to be useful. For binary networks, for instance, the marginal probabilities of $m$ nodes could be as small as $O(1/2^m)$, so we need $O(1/2^m)$ to be greater than $O((\ln N/N)^{(k+1)/2})$. This implies that $m < \ln\frac{1}{c} + \frac{(k+1)}{2}(\ln N - \ln\ln N)$ for some constant $c$. However, we shall see that our approximation is still useful for large $m$ as long as the quantity it tries to approximate is not too small.

For two-layer networks, an algorithm by Ng and Jordan (2000) yields a better error rate of $O(1/N^{(k+1)/2})$ by exploiting the Central Limit Theorem. However, this result is restricted to networks with only 2 layers. Barber and Sollich (1999) were also motivated by the Central Limit Theorem's effect to approximate $\eta_i^l$ by a multivariate Gaussian distribution, resulting in a similar exploitation of correlation between pairs of nodes in the parent layer as in our MF(2) approximation. Also related to Barber and Sollich's algorithm of using an approximating family of distribution is the assumed-density filtering approach (e.g., Minka, 2001). These approaches, however, do not provide an error bound guarantee.

### 4.3 Computing conditional expectations of nodes in different layers

For simplicity, in this subsection we shall consider binary layered networks. First, we are interested in the marginal probability of a fixed number of nodes in different layers. This can be expressed in terms of product of conditional probabilities of nodes in the same layer given values of nodes in the previous layer. As shown in the previous subsection, each of these conditional probabilities can be approximated with an error bound $O((\ln N/N)^{(k+1)/2})$ as $N \to \infty$, and the product can also be approximated with the same error bound.

Next, we consider approximating the probability of several nodes in the input layer conditioned on some nodes observed in the output layer $L$, i.e., $P(X_1^1 = x_1^1, \ldots, X_m^1 = x_m^1 | X_1^L = x_1^L, \ldots, X_n^L = x_n^L)$ for some fixed numbers $m$ and $n$ that are small compared to $N$. In a multi-layer network, when even one node in the output layer is observed, all nodes in the graph becomes dependent. Furthermore, the conditional probabilities of all nodes in the graph are generally not concentrated. Nevertheless, we can still approximate the conditional probability by approximating two marginal probabilities $P(X_1^1 = x_i^1, \ldots, X_m^1 = x_m^1, X_1^L = x_1^L, \ldots, X_n^L = x_n^L)$ and $P(X_1^L = x_1^L, \ldots, X_n^L = x_n^L)$ separately and taking the ratio. This boils down to the problem of computing the marginal probabilities of nodes residing in different layers of the graph. As discussed in the previous paragraph, since each marginal probabilities can be approximated with an asymptotic error bound $O((\ln N/N)^{(k+1)/2})$ as $N \to \infty$ (for binary networks), the same asymptotic error bound holds for the conditional probabilities of fixed number of nodes. In the next section, we shall present empirical results that show that this approximation is still quite good even when a large number of nodes are conditioned on.

## 5  Simulation results

In our experiments, we consider a large number of randomly generated multi-layer Bayesian networks with $L = 3$, $L = 4$ or $L = 5$ layers, and with the number of nodes in each layer ranging from 10 to 100. The number of parents of each node is chosen uniformly at random in $[2, N]$. We use the noisy-OR function for the local conditional probabilities; this choice has the advantage that we can obtain exact marginal probabilities for single nodes by exploiting the special structure of noisy-OR function (Heckerman,

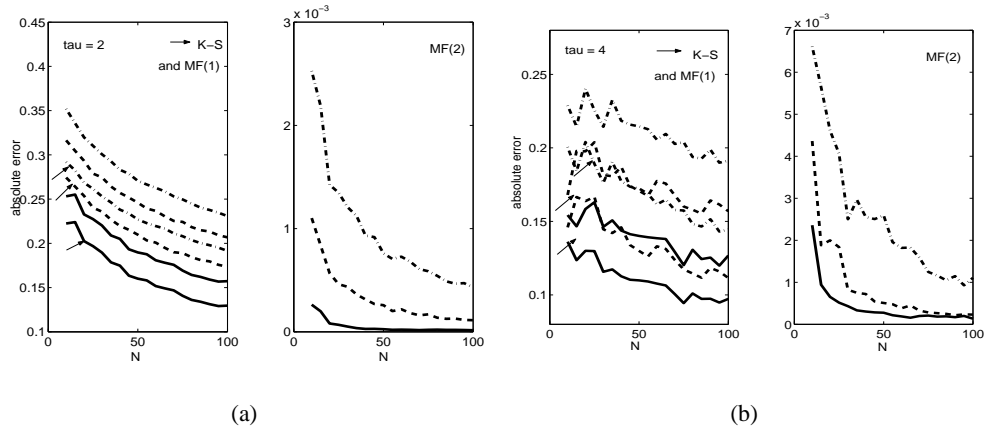

(a)                     (b)

Figure 1: The figures show the average error in the marginal probabilities of nodes in the output layer. The x-axis is the number of nodes in each layer ($N = 10, \ldots, 100$). The three curves (solid, dashed, dashdot) correspond to the different numbers of layers $L = 3, 4, 5$, respectively. Plot (a) corresponds to the case $\tau = 2$ and plot (b) corresponds to $\tau = 4$. In each pair of plots, the leftmost plot shows MF(1) and Kearns and Saul's algorithm (K-S) (with the latter being distinguished by black arrows), and the rightmost plot is MF(2). Note the scale on the y-axis for the rightmost plot is $10^{-3}$.

| k | 1 | 2 | 3 | 4 | 5 | 6 | 7 | 8 |
|---|---|---|---|---|---|---|---|---|
| Network 1 | 0.0001 | 0.0041 | 0.0052 | 0.0085 | 0.0162 | 0.0360 | 0.0738 | 0.1562 |
|  | 0.0007 | 0.0609 | 0.0912 | 0.1925 | 0.1862 | 0.3885 | 0.6262 | 1.6478 |
| Network 2 | 0.0003 | 0.0040 | 0.0148 | 0.0331 | 0.0981 | 0.1629 | 0.1408 | 0.1391 |
|  | 0.0018 | 0.0508 | 0.1431 | 0.3518 | 0.7605 | 0.7790 | 0.7118 | 0.9435 |
| Network 3 | 0.0002 | 0.0031 | 0.0082 | 0.0501 | 0.1095 | 0.0890 | 0.0957 | 0.1022 |
|  | 0.0008 | 0.0406 | 0.1150 | 0.6858 | 1.2392 | 0.6115 | 0.5703 | 0.7840 |

Table 1: The experiments were performed on 24-node networks (3 layers with $N = 8$ nodes in each layer). For each network, the first line shows the absolute error of our approximation of conditional probabilities of nodes in the input layer given values of the first $k$ nodes in the output layer, the second line shows the absolute error of the log likelihood of the $k$ nodes. The numbers were obtained by averaging over $k^2$ random instances of the $k$ nodes.

1989). All parameters $\theta_{ij}$ are uniformly distributed in $[0, \tau/N]$, with $\tau = 2$ and $\tau = 4$.

Figure 1 shows the error rates for computing the expectation of a single node in the output layer of the graph. The results for each $N$ are obtained by averaging over many graphical models with the same value of $N$. Our approximate algorithm, which is denoted by MF(2), runs fast: The running time for the largest network (with $L = 5, N = 100$) is approximately one minute.

We compare our algorithm (with $\gamma$ fixed to be $2/\alpha$) with that of Kearns and Saul (K-S). The MF(1) estimates are slightly worse that of the K-S algorithm, but they have the same error curve $O(\ln N/N)^{1/2}$. The MF(2) estimates, whose error curves were proven to be $O(\ln N/N)^{3/2}$, are better than both by orders of magnitude. The figure also shows that the error increases when we increase the size of the parameters (increase $\tau$).

Next, we consider the inference problem of computing conditional probabilities of the input layer given that the first $k$ nodes are observed in the output layer. We perform our experiments on several randomly generated three-layer networks with $N = 8$. This size allows us to be able to compute the conditional probabilities exactly.[2] For each value of

$k$, we generate $k^2$ samples of the observed nodes generated uniformly at random from the network and then compute the average of errors of conditional probability approximations. We observe that while the error of conditional probabilities is higher than those of marginal probabilities (see Table 1 and Figure 1), the error remains small despite the relatively large number of observed nodes $k$ compared to $N$.

## 6   Conclusions

We have presented a detailed analysis of concentration-of-expectation phenomena in layered Bayesian networks which use generalized linear models as local conditional probabilities. Our analysis encompasses a wide variety of probability distributions, including both discrete and continuous random variables. We also performed a large number of simulations in multi-layer network models, showing that our approach not only provides a useful theoretical analysis of concentration phenomena, but it also provides a fast and accurate inference algorithm for densely-connected multi-layer graphical models.

In the setting of Bayesian networks in which nodes have large in-degree, there are few viable options for probabilistic inference. Not only are junction tree algorithms infeasible, but (loopy) belief propagation algorithms are infeasible as well, because of the need to moralize. The mean-field algorithms that we have presented here are thus worthy of attention as one of the few viable methods for such graphs. As we have shown, the framework allows us to systematically trade time for accuracy with such algorithms, by accounting for interactions between neighboring nodes via the delta method.

**Acknowledgement.** We would like to thank Andrew Ng and Martin Wainwright for very useful discussions and feedback regarding this work.

## Footnotes

[1]The proofs of this and all other theorems can be found in a longer version of this paper, available at www.cs.berkeley.edu/~xuanlong.

[2] The amount of time spent on exact computation for each network is about 3 days, while our approximation routines take a few minutes.

## References

D. Barber and P. van de Laar, Variational cumulant expansions for intractable distributions. *Journal of Artificial Intelligence Research*, 10, 435-455, 1999.

L. Brown, *Fundamentals of Statistical Exponential Families with Applications in Statistical Decision Theory*, Institute of Mathematical Statistics, Hayward, CA, 1986.

P. McCullagh and J.A. Nelder, *Generalized Linear Models*, Chapman and Hall, London, 1983.

T. Minka, Expectation propagation for approximate Bayesian inference, In *Proc. UAI*, 2001.

D. Heckerman, A tractable inference algorithm for diagnosing multiple diseases, In *Proc. UAI*, 1989.

M.I. Jordan, Z. Ghahramani, T.S. Jaakkola and L.K. Saul, An introduction to variational methods for graphical models, In *Learning in Graphical Models*, Cambridge, MIT Press, 1998.

M.J. Kearns and L.K. Saul, Large deviation methods for approximate probabilistic inference, with rates of convergence, In *Proc. UAI*, 1998.

M.J. Kearns and L.K. Saul, Inference in multi-layer networks via large deviation bounds, *NIPS 11*, 1999.

A.Y. Ng and M.I. Jordan, Approximate inference algorithms for two-layer Baysian networks, *NIPS 12*, 2000.

D. Barber and P. Sollich, Gaussian fields for approximate inference in layered sigmoid belief networks, *NIPS 11*, 1999.

T. Plefka, Convergence condition of the TAP equation for the infinite-ranged Ising spin glass model, *J. Phys. A: Math. Gen., 15*(6), 1982.

J.S. Yedidia, W.T. Freeman, and Y. Weiss. Generalized belief propagation. *NIPS 13*, 2001.
